# Speaker Comparison with Inner Product Discriminant Functions

**W. M. Campbell**
MIT Lincoln Laboratory
Lexington, MA 02420
*wcampbell@ll.mit.edu*

**Z. N. Karam**
DSPG, MIT RLE, Cambridge MA
MIT Lincoln Laboratory, Lexington, MA
*zahi@mit.edu*

**D. E. Sturim**
MIT Lincoln Laboratory
Lexington, MA 02420
*sturim@ll.mit.edu*

## Abstract

Speaker comparison, the process of finding the speaker similarity between two speech signals, occupies a central role in a variety of applications—speaker verification, clustering, and identification. Speaker comparison can be placed in a geometric framework by casting the problem as a model comparison process. For a given speech signal, feature vectors are produced and used to adapt a Gaussian mixture model (GMM). Speaker comparison can then be viewed as the process of compensating and finding metrics on the space of adapted models. We propose a framework, inner product discriminant functions (IPDFs), which extends many common techniques for speaker comparison—support vector machines, joint factor analysis, and linear scoring. The framework uses inner products between the parameter vectors of GMM models motivated by several statistical methods. Compensation of nuisances is performed via linear transforms on GMM parameter vectors. Using the IPDF framework, we show that many current techniques are simple variations of each other. We demonstrate, on a 2006 NIST speaker recognition evaluation task, new scoring methods using IPDFs which produce excellent error rates and require significantly less computation than current techniques.

## 1   Introduction

Comparing speakers in speech signals is a common operation in many applications including forensic speaker recognition, speaker clustering, and speaker verification. Recent popular approaches to text-independent comparison include Gaussian mixture models (GMMs) [1], support vector machines [2, 3], and combinations of these techniques. When comparing two speech utterances, these approaches are used in a train and test methodology. One utterance is used to produce a model which is then scored against the other utterance. The resulting comparison score is then used to cluster, verify or identify the speaker.

Comparing speech utterances with kernel functions has been a common theme in the speaker recognition SVM literature [2, 3, 4]. The resulting framework has an intuitive geometric structure. Variable length sequences of feature vectors are mapped to a large dimensional SVM expansion vector. These vectors are "smoothed" to eliminate nuisances [2]. Then, a kernel function is applied to the

two vectors. The kernel function is an inner product which induces a metric on the set of vectors, so comparison is analogous to finding the distances between SVM expansion vectors.

A recent trend in the speaker recognition literature has been to move towards a more linear geometric view for non-SVM systems. Compensation via linear subspaces and supervectors of mean parameters of GMMs is presented in joint factor analysis [5]. Also, comparison of utterances via linear scoring is presented in [6]. These approaches have introduced many new ideas and perform well in speaker comparison tasks.

An unrealized effort in speaker recognition is to bridge the gap between SVMs and some of the new proposed GMM methods. One difficulty is that most SVM kernel functions in speaker comparison satisfy the Mercer condition. This restricts the scope of investigation of potential comparison strategies for two speaker utterances. Therefore, in this paper, we introduce the idea of inner product discriminant functions (IPDFs).

IPDFs are based upon the same basic operations as SVM kernel functions with some relaxation in structure. First, we map input utterances to vectors of fixed dimension. Second, we *compensate* the input feature vectors. Typically, this compensation takes the form of a linear transform. Third, we *compare* two compensated vectors with an inner product. The resulting comparison function is then used in an application specific way.

The focus of our initial investigations of the IPDF structure are the following. First, we show that many of the common techniques such as factor analysis, nuisance projection, and various types of scoring can be placed in the framework. Second, we systematically describe the various inner product and compensation techniques used in the literature. Third, we propose new inner products and compensation. Finally, we explore the space of possible combinations of techniques and demonstrate several novel methods that are computationally efficient and produce excellent error rates.

The outline of the paper is as follows. In Section 2, we describe the general setup for speaker comparison using GMMs. In Section 3, we introduce the IPDF framework. Section 4 explores inner products for the IPDF framework. Section 5 looks at methods for compensating for variability. In Section 6, we perform experiments on the NIST 2006 speaker recognition evaluation and explore different combinations of IPDF comparisons and compensations.

## 2  Speaker Comparison

A standard distribution used for text-independent speaker recognition is the Gaussian mixture model [1],

$$g(\mathbf{x}) = \sum_{i=1}^{N} \lambda_i \mathcal{N}(\mathbf{x}|\mathbf{m}_i, \Sigma_i). \tag{1}$$

Feature vectors are typically cepstral coefficients with associated smoothed first- and second-order derivatives.

We map a sequence of feature vectors, $\mathbf{x}_1^{N_x}$, from a speaker to a GMM by adapting a GMM universal background model (UBM). Here, we use the shorthand $\mathbf{x}_1^{N_x}$ to denote the sequence, $\mathbf{x}_1, \cdots, \mathbf{x}_{N_x}$. For the purpose of this paper, we will assume only the mixture weights, $\lambda_i$, and means, $\mathbf{m}_i$, in (1) are adapted. Adaptation of the means is performed with standard relevance MAP [1]. We estimate the mixture weights using the standard ML estimate. The adaptation yields new parameters which we stack into a parameter vector, $\mathbf{a}_x$, where

$$\mathbf{a}_x = \begin{bmatrix} \boldsymbol{\lambda}_x^t & \mathbf{m}_x^t \end{bmatrix}^t \tag{2}$$

$$= \begin{bmatrix} \lambda_{x,1} & \cdots & \lambda_{x,N} & \mathbf{m}_{x,1}^t & \cdots & \mathbf{m}_{x,N}^t \end{bmatrix}^t. \tag{3}$$

In speaker comparison, the problem is to compare two sequences of feature vectors, e.g., $\mathbf{x}_1^{N_x}$ and $\mathbf{y}_1^{N_y}$. To compare these two sequences, we adapt a GMM UBM to produce two sets of parameter vectors, $\mathbf{a}_x$ and $\mathbf{a}_y$, as in (2). The goal of our speaker comparison process can now be recast as a function that compares the two parameter vectors, $C(\mathbf{a}_x, \mathbf{a}_y)$, and produces a value that reflects the similarity of the speakers. Initial work in this area was performed using kernels from support vector machines [4, 7, 2], but we expand the scope to other types of discriminant functions.

## 3  Inner Product Discriminant Functions

The basic framework we propose for speaker comparison functions is composed of two parts—compensation and comparison. For compensation, the parameter vectors generated by adaptation in (2) can be transformed to remove nuisances or projected onto a speaker subspace. The second part of our framework is comparison. For the comparison of parameter vectors, we will consider natural distances that result in inner products between parameter vectors.

We propose the following inner product discriminant function (IPDF) framework for exploring speaker comparison,

$$C(\mathbf{a}_x, \mathbf{a}_y) = (L_x \mathbf{a}_x)^t D^2 (L_y \mathbf{a}_y) \tag{4}$$

where $L_x$, $L_y$ are linear transforms and potentially dependent on $\boldsymbol{\lambda}_x$ and/or $\boldsymbol{\lambda}_y$. The matrix $D$ is positive definite, usually diagonal, and possibly dependent on $\boldsymbol{\lambda}_x$ and/or $\boldsymbol{\lambda}_y$. Note, we also consider simple combinations of IPDFs to be in our framework—e.g., positively-weighted sums of IPDFs.

Several questions from this framework are: 1) what inner product gives the best speaker comparison performance, 2) what compensation strategy works best, 3) what tradeoffs can be made between accuracy and computational cost, and 4) how do the compensation and the inner product interact. We explore theoretical and experimental answers to these questions in the following sections.

## 4  Inner Products for IPDFs

In general, an inner product of the parameters should be based on a distance arising from a statistical comparison. We derive three straightforward methods in this section. We also relate some other methods, without being exhaustive, that fall in this framework that have been described in detail in the literature.

### 4.1  Approximate KL Comparison ($C_{\mathrm{KL}}$)

A straightforward strategy for comparing the GMM parameter vectors is to use an approximate form of the KL divergence applied to the induced GMM models. This strategy was used in [2] successfully with an approximation based on the log-sum inequality; i.e., for the GMMs, $g_x$ and $g_y$, with parameters $\mathbf{a}_x$ and $\mathbf{a}_y$,

$$D(g_x \| g_y) \leq \sum_{i=1}^{N} \lambda_{x,i} D \left( \mathcal{N}(\cdot; \mathbf{m}_{x,i}, \Sigma_i) \| \mathcal{N}(\cdot; \mathbf{m}_{y,i}, \Sigma_i) \right). \tag{5}$$

Here, $D(\cdot \| \cdot)$ is the KL divergence, and $\Sigma_i$ is from the UBM.

By symmetrizing (5) and substituting in the KL divergence between two Gaussian distributions, we obtain a distance, $d_s$, which upper bounds the symmetric KL divergence,

$$d_s(\mathbf{a}_x, \mathbf{a}_y) = D_s(\boldsymbol{\lambda}_x \| \boldsymbol{\lambda}_y) + \sum_{i=1}^{N} (0.5\lambda_{x,i} + 0.5\lambda_{y,i})(\mathbf{m}_{x,i} - \mathbf{m}_{y,i})^t \Sigma_i^{-1} (\mathbf{m}_{x,i} - \mathbf{m}_{y,i}). \tag{6}$$

We focus on the second term in (6) for this paper, but note that the first term could also be converted to a comparison function on the mixture weights. Using polarization on the second term, we obtain the inner product

$$C_{\mathrm{KL}}(\mathbf{a}_x, \mathbf{a}_y) = \sum_{i=1}^{N} (0.5\lambda_{x,i} + 0.5\lambda_{y,i}) \mathbf{m}_{x,i}^t \Sigma_i^{-1} \mathbf{m}_{y,i}. \tag{7}$$

Note that (7) can also be expressed more compactly as

$$C_{\mathrm{KL}}(\mathbf{a}_x, \mathbf{a}_y) = \mathbf{m}_x^t \left( (0.5\boldsymbol{\lambda}_x + 0.5\boldsymbol{\lambda}_y) \otimes I_n \right) \Sigma^{-1} \mathbf{m}_y \tag{8}$$

where $\Sigma$ is the block matrix with the $\Sigma_i$ on the diagonal, $n$ is the feature vector dimension, and $\otimes$ is the Kronecker product. Note that the non-symmetric form of the KL distance in (5) would result in the average mixture weights in (8) being replaced by $\boldsymbol{\lambda}_x$. Also, note that shifting the means by the UBM will not affect the distance in (6), so we can replace means in (8) by the UBM centered means.

## 4.2 GLDS kernel ($C_{\mathrm{GLDS}}$)

An alternate inner product approach is to use generalized linear discriminants and the corresponding kernel [4]. The overall structure of this GLDS kernel is as follows. A per feature vector expansion function is defined as

$$\mathbf{b}(\mathbf{x}_i) = [b_1(\mathbf{x}_i) \quad \cdots \quad b_m(\mathbf{x}_i)]^t . \tag{9}$$

The mapping between an input sequence, $\mathbf{x}_1^{N_x}$ is then defined as

$$\mathbf{x}_1^{N_x} \mapsto \mathbf{b}_x = \frac{1}{N_x} \sum_{i=1}^{N_x} \mathbf{b}(\mathbf{x}_i). \tag{10}$$

The corresponding kernel between two sequences is then

$$K_{\mathrm{GLDS}}(\mathbf{x}_1^{N_x}, \mathbf{y}_1^{N_y}) = \mathbf{b}_x^t \Gamma^{-1} \mathbf{b}_y \tag{11}$$

where

$$\Gamma = \frac{1}{N_z} \sum_{i=1}^{N_z} \mathbf{b}(\mathbf{z}_i)\mathbf{b}(\mathbf{z}_i)^t, \tag{12}$$

and $\mathbf{z}_1^{N_z}$ is a large set of feature vectors which is representative of the speaker population.

In the context of a GMM UBM, we can define an expansion as follows

$$\mathbf{b}(\mathbf{x}_i) = \left[ p(1|\mathbf{x}_i)(\mathbf{x}_i - \mathbf{m}_1)^t \quad \cdots \quad p(N|\mathbf{x}_i)(\mathbf{x}_i - \mathbf{m}_N)^t \right]^t \tag{13}$$

where $p(j|\mathbf{x}_i)$ is the posterior probability of mixture component $j$ given $\mathbf{x}_i$, and $\mathbf{m}_j$ is from a UBM. Using (13) in (10), we see that

$$\mathbf{b}_x = (\boldsymbol{\lambda}_x \otimes I_n)(\mathbf{m}_x - \mathbf{m}) \text{ and } \mathbf{b}_y = (\boldsymbol{\lambda}_y \otimes I_n)(\mathbf{m}_y - \mathbf{m}) \tag{14}$$

where $\mathbf{m}$ is the stacked means of the UBM. Thus, the GLDS kernel inner product is

$$C_{\mathrm{GLDS}}(\mathbf{a}_x, \mathbf{a}_y) = (\mathbf{m}_x - \mathbf{m})^t (\boldsymbol{\lambda}_x \otimes I_n)\Gamma^{-1}(\boldsymbol{\lambda}_y \otimes I_n)(\mathbf{m}_y - \mathbf{m}). \tag{15}$$

Note that $\Gamma$ in (12) is almost the UBM covariance matrix, but is not quite the same because of a squaring of the $p(j|\mathbf{z}_i)$ in the diagonal. As is commonly assumed, we will consider a diagonal approximation of $\Gamma$, see [4].

### 4.3 Gaussian-Distributed Vectors

A common assumption in the factor analysis literature [5] is that the parameter vector $\mathbf{m}_x$ as $x$ varies has a Gaussian distribution. If we assume a single covariance for the entire space, then the resulting likelihood ratio test between two Gaussian distributions results in a linear discriminant [8].

More formally, suppose that we have a distribution with mean $\mathbf{m}_x$ and we are trying to distinguish from a distribution with the UBM mean $\mathbf{m}$, then the discriminant function is [8],

$$h(\mathbf{x}) = (\mathbf{m}_x - \mathbf{m})^t \Upsilon^{-1}(\mathbf{x} - \mathbf{m}) + c_x \tag{16}$$

where $c_x$ is a constant that depends on $\mathbf{m}_x$, and $\Upsilon$ is the covariance in the parameter vector space. We will assume that the comparison function can be normalized (e.g., by Z-norm [1]), so that $c_x$ can be dropped. We now apply the discriminant function to another mean vector, $\mathbf{m}_y$, and obtain the following comparison function

$$C_G(\mathbf{a}_x, \mathbf{a}_y) = (\mathbf{m}_x - \mathbf{m})^t \Upsilon^{-1}(\mathbf{m}_y - \mathbf{m}). \tag{17}$$

### 4.4 Other Methods

Several other methods are possible for comparing the parameter vectors that arise either from ad hoc methods or from work in the literature. We describe a few of these in this section.

**Geometric Mean Comparison** ($C_{\mathrm{GM}}$). A simple symmetric function that is similar to the KL (8) and GLDS (15) comparison functions is arrived at by replacing the arithmetic mean in $C_{\mathrm{KL}}$ by a geometric mean. The resulting kernel is

$$C_{GM}(\mathbf{a}_x, \mathbf{a}_y) = (\mathbf{m}_x - \mathbf{m})^t (\boldsymbol{\lambda}_x^{1/2} \otimes I_n)\Sigma^{-1}(\boldsymbol{\lambda}_y^{1/2} \otimes I_n)(\mathbf{m}_y - \mathbf{m}) \tag{18}$$

where $\Sigma$ is the block diagonal UBM covariances.

**Fisher Kernel** ($C_F$). The Fisher kernel specialized to the UBM case has several forms [3]. The main variations are the choice of covariance in the inner product and the choice of normalization of the gradient term. We took the best performing configuration for this paper—we normalize the gradient by the number of frames which results in a mixture weight scaling of the gradient. We also use a diagonal data-trained covariance term. The resulting comparison function is

$$C_F(\mathbf{a}_x, \mathbf{a}_y) = \left[(\boldsymbol{\lambda}_x \otimes I_n)\Sigma^{-1}(\mathbf{m}_x - \mathbf{m})\right]^t \Phi^{-1} \left[(\boldsymbol{\lambda}_y \otimes I_n)\Sigma^{-1}(\mathbf{m}_y - \mathbf{m})\right] \qquad (19)$$

where $\Phi$ is a diagonal matrix acting as a variance normalizer.

**Linearized Q-function** ($C_Q$). Another form of inner product may be derived from the linear Q-scoring shown in [6]. In this case, the scoring is given as $(\mathbf{m}_{\text{train}} - \mathbf{m})^t \Sigma^{-1}(\mathbf{F} - \mathbf{Nm})$ where $\mathbf{N}$ and $\mathbf{F}$ are the zeroth and first order sufficient statistics of a test utterance, $\mathbf{m}$ is the UBM means, $\mathbf{m}_{\text{train}}$ is the mean of a training model, and $\Sigma$ is the block diagonal UBM covariances. A close approximation of this function can be made by using a small relevance factor in MAP adaptation of the means to obtain the following comparison function

$$C_Q(\mathbf{a}_x, \mathbf{a}_y) = (\mathbf{m}_x - \mathbf{m})^t \Sigma^{-1}(\boldsymbol{\lambda}_y \otimes I_n)(\mathbf{m}_y - \mathbf{m}). \qquad (20)$$

Note that if we symmetrize $C_Q$, this gives us $C_{\text{KL}}$; this analysis ignores for a moment that in [6], compensation is also asymmetric.

**KL Kernel** ($K_{\text{KL}}$). By assuming the mixture weights are constant and equal to the UBM mixture in the comparison function $C_{\text{KL}}$ (7), we obtain the KL kernel,

$$K_{KL}(\mathbf{m}_x, \mathbf{m}_y) = \mathbf{m}_x^t \left(\boldsymbol{\lambda} \otimes I_n\right) \Sigma^{-1}\mathbf{m}_y \qquad (21)$$

where $\boldsymbol{\lambda}$ are the UBM mixture weights. This kernel has been used extensively in SVM speaker recognition [2].

An analysis of the different inner products in the preceding sections shows that many of the methods presented in the literature have a similar form, but are interestingly derived with quite disparate techniques. Our goal in the experimental section is to understand how these comparison function perform and how they interact with compensation.

## 5   Compensation in IPDFs

Our next task is to explore compensation methods for IPDFs. Our focus will be on subspace-based methods. With these methods, the fundamental assumption is that either speakers and/or nuisances are confined to a small subspace in the parameter vector space. The problem is to use this knowledge to produce a higher signal (speaker) to noise (nuisance) representation of the speaker. Standard notation is to use $U$ to represent the nuisance subspace and to have $V$ represent the speaker subspace. Our goal in this section is to recast many of the methods in the literature in a standard framework with oblique and orthogonal projections.

To make a cohesive presentation, we introduce some notation. We define an orthogonal projection with respect to a metric, $P_{U,D}$, where $D$ and $U$ are full rank matrices as

$$P_{U,D} = U(U^t D^2 U)^{-1}U^t D^2 \qquad (22)$$

where $DU$ is a linearly independent set, and the metric is $\|x - y\|_D = \|Dx - Dy\|_2$. The process of projection, e.g. $y = P_{U,D}b$, is equivalent to solving the least-squares problem, $\hat{x} = \arg\min_x \|Ux - b\|_D$ and letting $y = U\hat{x}$. For convenience, we also define the projection onto the orthogonal complement of $U$, $U^\perp$, as $Q_{U,D} = P_{U^\perp,D} = I - P_{U,D}$. Note that we can regularize the projection $P_{U,D}$ by adding a diagonal term to the inverse in (22); the resulting operation remains linear but is no longer a projection.

We also define the oblique projection onto $V$ with null space $U + (U + V)^\perp$ and metric induced by $D$. Let $QR$ be the (skinny) QR decomposition of the matrix $[UV]$ in the $D$ norm (i.e., $Q^t D^2 Q = I$), and $Q_V$ be the columns corresponding to $V$ in the matrix $Q$. Then, the oblique (non-orthogonal) projection onto $V$ is

$$O_{V,U,D} = V(Q_V^t D^2 V)^{-1}Q_V^t D^2. \qquad (23)$$

The use of projections in our development will add geometric understanding to the process of compensation.

## 5.1 Nuisance Attribute Projection (NAP)

A framework for eliminating nuisances in the parameter vector based on projection was shown in [2]. The basic idea is to assume that nuisances are confined to a small subspace and can be removed via an orthogonal projection, $\mathbf{m}_x \mapsto Q_{U,D}\mathbf{m}_x$. One justification for using subspaces comes from the perspective that channel classification can be performed with inner products along one-dimensional subspaces. Therefore, the projection removes channel specific directions from the parameter space.

The NAP projection uses the metric induced by a kernel in an SVM. For the GMM context, the standard kernel used is the approximate KL comparison (8) [2]. We note that since $D$ is known *a priori* to speaker comparison, we can orthonormalize the matrix $DU$ and apply the projection as a matrix multiply. The resulting projection has $D = \left( \boldsymbol{\lambda}^{1/2} \otimes I_n \right) \Sigma^{-1/2}$.

## 5.2 Factor Analysis and Joint Factor Analysis

The joint factor analysis (JFA) model assumes that the mean parameter vector can be expressed as

$$\mathbf{m}_{s,\text{sess}} = \mathbf{m} + U\mathbf{x} + V\mathbf{y} \tag{24}$$

where $\mathbf{m}_{s,\text{sess}}$ is the speaker- and session-dependent mean parameter vector, $U$ and $V$ are matrices with small rank, and $\mathbf{m}$ is typically the UBM. Note that for this section, we will use the standard variables for factor analysis, $\mathbf{x}$ and $\mathbf{y}$, even though they conflict with our earlier development. The goal of joint factor analysis is to find solutions to the latent variables $\mathbf{x}$ and $\mathbf{y}$ given training data. In (24), the matrix $U$ represents a nuisance subspace, and $V$ represents a speaker subspace. Existing work on this approach for speaker recognition uses both maximum likelihood (ML) estimates and MAP estimates of $\mathbf{x}$ and $\mathbf{y}$ [9, 5]. In the latter case, a Gaussian prior with zero mean and diagonal covariance for $\mathbf{x}$ and $\mathbf{y}$ is assumed. For our work, we focus on the ML estimates [9] of $\mathbf{x}$ and $\mathbf{y}$ in (24), since we did not observe substantially different performance from MAP estimates in our experiments.

Another form of modeling that we will consider is factor analysis (FA). In this case, the term $V\mathbf{y}$ is replaced by a constant vector representing the true speaker model, $\mathbf{m}_s$; the goal is then to estimate $\mathbf{x}$. Typically, as a simplification, $\mathbf{m}_s$ is assumed to be zero when calculating sufficient statistics for estimation of $\mathbf{x}$ [10].

The solution to both JFA and FA can be unified. For the JFA problem, if we stack the matrices $[UV]$, then the problem reverts to the FA problem. Therefore, we initially study the FA problem. Note that we also restrict our work to only one EM iteration of the estimation of the factors, since this strategy works well in practice.

The standard ML solution to FA [9] for one EM iteration can be written as

$$\left[ U^t \Sigma^{-1} (\mathbf{N} \otimes I_n) U \right] \mathbf{x} = U^t \Sigma^{-1} \left[ \mathbf{F} - (\mathbf{N} \otimes I_n) \mathbf{m} \right] \tag{25}$$

where $\mathbf{F}$ is the vector of first order sufficient statistics, and $\mathbf{N}$ is the diagonal matrix of zeroth order statistics (expected counts). The sufficient statistics are obtained from the UBM applied to an input set of feature vectors. We first let $N_{\text{t}} = \sum_{i=1}^{N} N_i$ and multiply both sides of (25) by $1/N_t$. Now use relevance MAP with a small relevance factor and $\mathbf{F}$ and $\mathbf{N}$ to obtain $\mathbf{m}_s$; i.e., both $\mathbf{m}_s - \mathbf{m}$ and $\mathbf{F} - (\mathbf{N} \otimes I_n)\mathbf{m}$ will be nearly zero in the entries corresponding to small $N_i$. We obtain

$$\left[ U^t \Sigma^{-1} (\boldsymbol{\lambda}_s \otimes I_n) U \right] \mathbf{x} = U^t \Sigma^{-1} (\boldsymbol{\lambda}_s \otimes I_n) \left[ \mathbf{m}_s - \mathbf{m} \right] \tag{26}$$

where $\boldsymbol{\lambda}_s$ is the speaker dependent mixture weights. We note that (26) are the normal equations for the least-squares problem, $\hat{\mathbf{x}} = \operatorname{argmin}_{\mathbf{x}} \|U\mathbf{x} - (\mathbf{m}_s - \mathbf{m})\|_D$ where $D$ is given below. This solution is not unexpected since ML estimates commonly lead to least-squares problems with GMM distributed data [11].

Once the solution to (26) is obtained, the resulting $U\mathbf{x}$ is subtracted from an estimate of the speaker mean, $\mathbf{m}_s$ to obtain the compensated mean. If we assume that $\mathbf{m}_s$ is obtained by a relevance map adaptation from the statistics $\mathbf{F}$ and $\mathbf{N}$ with a small relevance factor, then the FA process is well approximated by

$$\mathbf{m}_s \mapsto Q_{U,D}\mathbf{m}_s \tag{27}$$

where

$$D = \left( \boldsymbol{\lambda}_s^{1/2} \otimes I_n \right) \Sigma^{-1/2}. \tag{28}$$

JFA becomes an extension of the FA process we have demonstrated. One first projects onto the stacked $UV$ space. Then another projection is performed to eliminate the $U$ component of variability. This can be expressed as a single oblique projection; i.e., the JFA process is

$$\mathbf{m}_s \mapsto O_{V,U,I} P_{[UV],D} \mathbf{m}_s = O_{V,U,D} \mathbf{m}_s. \tag{29}$$

### 5.3 Comments and Analysis

Several comments should be made on compensation schemes and their use in speaker comparison. First, although NAP and ML FA (27) were derived in substantially different ways, they are essentially the same operation, an orthogonal projection. The main difference is in the choice of metrics under which they were originally proposed. For NAP, the metric depends on the UBM only, and for FA it is utterance and UBM dependent.

A second observation is that the JFA oblique projection onto $V$ has substantially different properties than a standard orthogonal projection. When JFA is used in speaker recognition [5, 6], typically JFA is performed in training, but the test utterance is compensated only with FA. In our notation, applying JFA with linear scoring [6] gives

$$C_Q(O_{V,U,D_1} \mathbf{m}_1, Q_{U,D_2} \mathbf{m}_2) \tag{30}$$

where $\mathbf{m}_1$ and $\mathbf{m}_2$ are the mean parameter vectors estimated from the training and testing utterances, respectively; also, $D_1 = (\boldsymbol{\lambda}_1^{1/2} \otimes I_n) \Sigma^{-1/2}$ and $D_2 = (\boldsymbol{\lambda}_2^{1/2} \otimes I_n) \Sigma^{-1/2}$. Our goal in the experiments section is to disentangle and understand some of the properties of scoring methods such as (30). What is significant in this process—mismatched train/test compensation, data-dependent metrics, or asymmetric scoring?

A final note is that training the subspaces for the various projections *optimally* is not a process that is completely understood. One difficulty is that the metric used for the inner product may not correspond to the metric for compensation. As a baseline, we used the same subspace for all comparison functions. The subspace was obtained with an ML style procedure for training subspaces similar to [11] but specialized to the factor analysis problem as in [5].

## 6   Speaker Comparison Experiments

Experiments were performed on the NIST 2006 speaker recognition evaluation (SRE) data set. Enrollment/verification methodology and the evaluation criterion, equal error rate (EER) and minDCF, were based on the NIST SRE evaluation plan [12]. The main focus of our efforts was the one conversation enroll, one conversation verification task for telephone recorded speech. T-Norm models and Z-Norm [13] speech utterances were drawn from the NIST 2004 SRE corpus. Results were obtained for both the English only task (Eng) and for all trials (All) which includes speakers that enroll/verify in different languages.

Feature extraction was performed using HTK [14] with 20 MFCC coefficients, deltas, and acceleration coefficients for a total of 60 features. A GMM UBM with 512 mixture components was trained using data from NIST SRE 2004 and from Switchboard corpora. The dimension of the nuisance subspace, $U$, was fixed at 100; the dimension of the speaker space, $V$, was fixed at 300.

Results are in Table 1. In the table, we use the following notation,

$$D_{\text{UBM}} = \left(\boldsymbol{\lambda}^{1/2} \otimes I_n\right) \Sigma^{-1/2}, \; D_1 = \left(\boldsymbol{\lambda}_1^{1/2} \otimes I_n\right) \Sigma^{-1/2}, \; D_2 = \left(\boldsymbol{\lambda}_2^{1/2} \otimes I_n\right) \Sigma^{-1/2} \tag{31}$$

where $\boldsymbol{\lambda}$ are the UBM mixture weights, $\boldsymbol{\lambda}_1$ are the mixture weights estimated from the enrollment utterance, and $\boldsymbol{\lambda}_2$ are the mixture weights estimated from the verification utterance. We also use the notation $D_L$, $D_G$, and $D_F$ to denote the parameters of the metric for the GLDS, Gaussian, and Fisher comparison functions from Sections 4.2, 4.3, and 4.4, respectively.

An analysis of the results in Table 1 shows several trends. First, the performance of the best IPDF configurations is as good or better than the state of the art SVM and JFA implementations. Second, the compensation method that dominates good performance is an orthogonal complement of the nuisance subspace, $Q_{U,D}$. Combining a nuisance projection with an oblique projection is fine, but

Table 1: A comparison of baseline systems and different IPDF implementations

| Comparison Function | Enroll Comp. | Verify Comp. | EER All (%) | minDCF All (×100) | EER Eng (%) | minDCF Eng (×100) |
|---|---|---|---|---|---|---|
| Baseline SVM | $Q_{U,D_{\mathrm{UBM}}}$ | $Q_{U,D_{\mathrm{UBM}}}$ | 3.82 | 1.82 | 2.62 | 1.17 |
| Baseline JFA, $C_Q$ | $O_{V,U,D_1}$ | $Q_{U,D_2}$ | 3.07 | 1.57 | 2.11 | 1.23 |
| $C_{KL}$ | $O_{V,U,D_1}$ | $Q_{U,D_2}$ | 3.21 | 1.70 | 2.32 | 1.32 |
| $C_{KL}$ | $O_{V,U,D_1}$ | $O_{V,U,D_2}$ | 8.73 | 5.06 | 8.06 | 4.45 |
| $C_{KL}$ | $Q_{U,D_1}$ | $Q_{U,D_2}$ | 2.93 | 1.55 | 1.89 | 0.93 |
| $C_{KL}$ | $Q_{U,D_{\mathrm{UBM}}}$ | $Q_{U,D_{\mathrm{UBM}}}$ | 3.03 | 1.55 | 1.92 | 0.95 |
| $C_{KL}$ | $I - O_{U,V,D_1}$ | $I - O_{U,V,D_2}$ | 7.10 | 3.60 | 6.49 | 3.13 |
| $C_{GM}$ | $Q_{U,D_1}$ | $Q_{U,D_2}$ | 2.90 | 1.59 | 1.73 | 0.98 |
| $C_{GM}$ | $Q_{U,D_{\mathrm{UBM}}}$ | $Q_{U,D_{\mathrm{UBM}}}$ | 3.01 | 1.66 | 1.89 | 1.05 |
| $C_{GM}$ | $Q_{U,D_{\mathrm{UBM}}}$ | $I$ | 3.95 | 1.93 | 2.76 | 1.26 |
| $K_{KL}$ | $Q_{U,D_{\mathrm{UBM}}}$ | $Q_{U,D_{\mathrm{UBM}}}$ | 4.95 | 2.46 | 3.73 | 1.75 |
| $K_{KL}$ | $Q_{U,D_1}$ | $Q_{U,D_2}$ | 5.52 | 2.85 | 4.43 | 2.15 |
| $C_{GLDS}$ | $Q_{U,D_L}$ | $Q_{U,D_L}$ | 3.60 | 1.93 | 2.27 | 1.23 |
| $C_G$ | $Q_{U,D_G}$ | $Q_{U,D_G}$ | 5.07 | 2.52 | 3.89 | 1.87 |
| $C_F$ | $Q_{U,D_F}$ | $Q_{U,D_F}$ | 3.56 | 1.89 | 2.22 | 1.12 |

Table 2: Summary of some IPDF performances and computation time normalized to a baseline system. Compute time includes compensation and inner product only.

| Comparison Function | Enroll Comp. | Verify Comp. | EER Eng (%) | minDCF Eng (×100) | Compute time |
|---|---|---|---|---|---|
| $C_Q$ | $O_{V,U,D_1}$ | $Q_{U,D_2}$ | 2.11 | 1.23 | 1.00 |
| $C_{GM}$ | $Q_{U,D_1}$ | $Q_{U,D_2}$ | 1.73 | 0.98 | 0.17 |
| $C_{GM}$ | $Q_{U,D_{\mathrm{UBM}}}$ | $Q_{U,D_{\mathrm{UBM}}}$ | 1.89 | 1.05 | 0.08 |
| $C_{GM}$ | $Q_{U,D_{\mathrm{UBM}}}$ | $I$ | 2.76 | 1.26 | 0.04 |

using only oblique projections onto V gives high error rates. A third observation is that comparison functions whose metrics incorporate $\boldsymbol{\lambda}_1$ and $\boldsymbol{\lambda}_2$ perform significantly better than ones with fixed $\boldsymbol{\lambda}$ from the UBM. In terms of best performance, $C_{KL}$, $C_Q$, and $C_{GM}$ perform similarly. For example, the $95\%$ confidence interval for $2.90\%$ EER is $[2.6, 3.3]\%$.

We also observe that a nuisance projection with fixed $D_{\mathrm{UBM}}$ gives similar performance to a projection involving a "variable" metric, $D_i$. This property is fortuitous since a fixed projection can be precomputed and stored and involves significantly reduced computation. Table 2 shows a comparison of error rates and compute times normalized by a baseline system. For the table, we used precomputed data as much as possible to minimize compute times. We see that with an order of magnitude reduction in computation and a significantly simpler implementation, we can achieve the same error rate.

# 7 Conclusions and future work

We proposed a new framework for speaker comparison, IPDFs, and showed that several recent systems in the speaker recognition literature can be placed in this framework. We demonstrated that using mixture weights in the inner product is the key component to achieve significant reductions in error rates over a baseline SVM system. We also showed that elimination of the nuisance subspace via an orthogonal projection is a computationally simple and effective method of compensation. Most effective methods of compensation in the literature (NAP, FA, JFA) are straightforward variations of this idea. By exploring different IPDFs using these insights, we showed that computation can be reduced substantially over baseline systems with similar accuracy to the best performing systems. Future work includes understanding the performance of IPDFs for different tasks, incorporating them into an SVM system, and hyperparameter training.

## Footnotes

*This work was sponsored by the Federal Bureau of Investigation under Air Force Contract FA8721-05-C-0002. Opinions, interpretations, conclusions, and recommendations are those of the authors and are not necessarily endorsed by the United States Government.

# References

[1] Douglas A. Reynolds, T. F. Quatieri, and R. Dunn, "Speaker verification using adapted Gaussian mixture models," *Digital Signal Processing*, vol. 10, no. 1-3, pp. 19–41, 2000.

[2] W. M. Campbell, D. E. Sturim, D. A. Reynolds, and A. Solomonoff, "SVM based speaker verification using a GMM supervector kernel and NAP variability compensation," in *Proc. ICASSP*, 2006, pp. I97–I100.

[3] C. Longworth and M. J. F. Gales, "Derivative and parametric kernels for speaker verification," in *Proc. Interspeech*, 2007, pp. 310–313.

[4] W. M. Campbell, "Generalized linear discriminant sequence kernels for speaker recognition," in *Proc. ICASSP*, 2002, pp. 161–164.

[5] P. Kenny, P. Ouellet, N. Dehak, V. Gupta, and P. Dumouchel, "A study of inter-speaker variability in speaker verification," *IEEE Transactions on Audio, Speech and Language Processing*, 2008.

[6] Ondrej Glembek, Lukas Burget, Najim Dehak, Niko Brummer, and Patrick Kenny, "Comparison of scoring methods used in speaker recognition with joint factor analysis," in *Proc. ICASSP*, 2009.

[7] Pedro J. Moreno, Purdy P. Ho, and Nuno Vasconcelos, "A Kullback-Leibler divergence based kernel for SVM classification in multimedia applications," in *Adv. in Neural Inf. Proc. Systems 16*, S. Thrun, L. Saul, and B. Schölkopf, Eds. MIT Press, Cambridge, MA, 2004.

[8] Keinosuke Fukunaga, *Introduction to Statistical Pattern Recognition*, Academic Press, 1990.

[9] Simon Lucey and Tsuhan Chen, "Improved speaker verification through probabilistic subspace adaptation," in *Proc. Interspeech*, 2003, pp. 2021–2024.

[10] Robbie Vogt, Brendan Baker, and Sridha Sriharan, "Modelling session variability in text-independent speaker verification," in *Proc. Interspeech*, 2005, pp. 3117–3120.

[11] Mark J. F. Gales, "Cluster adaptive training of hidden markov models," *IEEE Trans. Speech and Audio Processing*, vol. 8, no. 4, pp. 417–428, 2000.

[12] M. A. Przybocki, A. F. Martin, and A. N. Le, "NIST speaker recognition evaluations utilizing the Mixer corpora—2004,2005,2006," *IEEE Trans. on Speech, Audio, Lang.*, vol. 15, no. 7, pp. 1951–1959, 2007.

[13] Roland Auckenthaler, Michael Carey, and Harvey Lloyd-Thomas, "Score normalization for text-independent speaker verification systems," *Digital Signal Processing*, vol. 10, pp. 42–54, 2000.

[14] J. Odell, D. Ollason, P. Woodland, S. Young, and J. Jansen, *The HTK Book for HTK V2.0*, Cambridge University Press, Cambridge, UK, 1995.

